# Dip-means: an incremental clustering method for estimating the number of clusters

**Argyris Kalogeratos**
Department of Computer Science
University of Ioannina
Ioannina, Greece 45110
akaloger@cs.uoi.gr

**Aristidis Likas**
Department of Computer Science
University of Ioannina
Ioannina, Greece 45110
arly@cs.uoi.gr

## Abstract

Learning the number of clusters is a key problem in data clustering. We present dip-means, a novel robust incremental method to learn the number of data clusters that can be used as a wrapper around any iterative clustering algorithm of k-means family. In contrast to many popular methods which make assumptions about the underlying cluster distributions, dip-means only assumes a fundamental cluster property: each cluster to admit a unimodal distribution. The proposed algorithm considers each cluster member as an individual 'viewer' and applies a univariate statistic hypothesis test for unimodality (dip-test) on the distribution of distances between the viewer and the cluster members. Important advantages are: i) the unimodality test is applied on univariate distance vectors, ii) it can be directly applied with kernel-based methods, since only the pairwise distances are involved in the computations. Experimental results on artificial and real datasets indicate the effectiveness of our method and its superiority over analogous approaches.

## 1 Introduction

Data clustering is a data analysis methodology which aims to automatically reveal the underlying structure of data. It produces a partition of a given dataset into $k$ groups of similar objects and as a task is widely applicable in artificial intelligence, data mining, statistics and other information processing fields. Although it is an NP-hard problem, various algorithms can find reasonable clusterings in polynomial time. Most clustering methods consider the number of clusters $k$ as a required input, and then they apply an optimization procedure to adjust the parameters of the assumed cluster model. As a consequence, in exploratory analysis, where the data characteristics are not known in advance, an appropriate $k$ value must be chosen. This is a rather difficult problem, but at the same time very fundamental in order to apply data clustering in practice.

Several algorithms have been proposed to determine a proper $k$ value, most of which wrap around an iterative model-based clustering framework, such as the k-means or the more general Expectation-Maximization (EM). In a top-down (incremental) strategy they start with one cluster and proceed to splitting as long as a certain criterion is satisfied. At each phase, they evaluate the clustering produced with a fixed $k$ and they decide whether to increase the number of clusters as follows:

Repeat until no changes occur in the model structure
1. Improve model parameters by running a conventional clustering algorithm for a fixed $k$ value.
2. Improve model structure, usually through cluster splitting.

One of the first attempts in extending k-means in this direction was x-means [1] which uses a regularization penalty based on model's complexity. To this end, Bayesian Information Criterion (BIC) [2] was used, and among many models the one with highest BIC is selected. This criterion works

well only in cases where there are plenty of data and well-separated spherical clusters. Alternative selection criteria have also been examined in literature [3].

G-means [4] is another extension to k-means that uses a statistical test for the hypothesis that each cluster has been generated from Gaussian distribution. Since statistical tests become weaker in high dimensions, the algorithm first projects the datapoints of a cluster on an axis of high variance and then applies Anderson-Darling statistic with a fixed significance level $\alpha$. Clusters that are not accepted are split repeatedly until the entire assumed mixture of Gaussians is discovered. Projected g-means (pg-means) [5] again assumes that the dataset has been generated from a Gaussian mixture, but it tests the overall model at once and not each cluster separately. Pg-means bases on the EM algorithm. Using a series of random linear projections, it constructs a one-dimensional projection of the dataset and the learned model and then tests the model fitness in the projected space with Kolmogorov-Smirnov (KS) test. The advantage of this method is the ability to discover Gaussian clusters of various scales and different covariances, that may overlap. Bayesian k-means [6] introduces Maximization-Expectation (ME) to learn a mixture model by maximizing over hidden variables (datapoint assignments to clusters) and computing expectation over random model parameters (centers and covariances). If the data come from a mixture of Gaussian components, this method can be used to find the correct number of clusters and is competitive to the aforementioned approaches. Other alternatives have also been proposed, such as gap statistic [7], self-tuning spectral clustering [8], data spectroscopic clustering [9], and stability-based model validation [10]-[12], however they are not closely related to the proposed method.

Our work is primarily motivated by the non generality of the approaches in [4] and [5], as they make Gausssianity assumptions about the underlying data distribution. As a consequence, they tend to overfit for clusters that are uniformly distributed, or have a non-Gaussian unimodal distribution. Additional limitations are that they are designed to handle numerical vectors only and require the data in the original dataspace. The contribution of our work is two-fold. Firstly, we propose a statistical test for unimodality, called *dip-dist*, to be applied into a data subset in order to determine if it contains a single or multiple cluster structures. Thus, we make a more general assumption about what is an acceptable cluster. Moreover, the test involves pairwise distances or similarities and not the original data vectors. Secondly, we propose the *dip-means* incremental clustering method which is a wrapper around k-means. We experimentally show that dip-means is able to cope with datasets containing clusters of arbitrary density distributions. Moreover, it can be easily extended in kernel space by using the kernel k-means [13] and modifying appropriately the cluster splitting procedure.

## 2   Dip-dist criterion for cluster structure evaluation

In cluster analysis, the detection of multiple cluster structures in a dataset requires assumptions about what the clusters we seek look like. The assumptions about the presence of certain data characteristics along with the tests employed for verification, considerably influence the performance of various methods. It is highly desirable for the assumptions to be general in order not to restrict the applicability of the method to certain types of clusters only (e.g. Gaussian). Moreover, it is of great value for a method to be able to verify the assumed cluster hypothesis with well designed statistical hypothesis tests that are theoretically sound, in contrast to various alternative ad hoc criteria.

We propose the novel *dip-dist criterion* for evaluating the cluster structure of a dataset that is based on testing the empirical density distribution of the data for unimodality. The *unimodality assumption* implies that the empirical density of an acceptable cluster should have a single mode; a region where the density becomes maximum, while non-increasing density is observed when moving away from the mode. There are no other underlying assumptions about the shape of a cluster and the distribution that generated the empirically observed unimodal property. Under this assumption, it is possible to identify clusters generated by various unimodal distributions, such as Gaussian, Student-t, etc. The Uniform distribution can also be identified, since it is an extreme single mode case where the mode covers all the region with non-zero density.

A convenient issue is that unimodality can be verified using powerful statistical hypothesis tests (especially for one-dimensional data), such as Silverman's method which uses fixed-width kernel density estimates [14] or the widely used Hartigan's dip statistic [15]. As the dimensionality of the data increases, the tests require a sufficient number of data points in order to be reliable. Thus, although the data may be of arbitrary dimensionality, it is important to apply unimodality tests on

one-dimensional data values. Furthermore, it would be desirable, if the test could also be applied in cases where the distance (or similarity) matrix is given and not the original datapoints.

To meet the above requirements we propose the *dip-dist* criterion for determining unimodality in a set of datapoints using only their pairwise distances (or similarities). More specifically, if we consider an arbitrary datapoint as a *viewer* and form a vector whose components are the distances of the viewer from all the datapoints, then the distribution of the values in this distance vector could reveal information about the cluster structure. In presence of a single cluster, the distribution of distances is expected to be unimodal. In the case of two distinct clusters, the distribution of distances should exhibit two distinct modes, with each mode containing the distances to the datapoints of each cluster. Consequently, a unimodality test on the distribution of the values of the distance vector would provide indication about the unimodality of the cluster structure. However, there is a dependence of the results on the selected viewer. Intuitively, viewers at the boundaries of the set are expected to form distance vectors whose density modes are more distinct in case of more than one clusters. To tackle the viewer selection problem, we consider all the datapoints of the set as individual viewers and perform the unimodality test on the distance vector of each viewer. If there exist viewers that reject unimodality (called *split viewers*), we conclude that the examined cluster includes multiple cluster structures.

For testing unimodality we use Hartigans' dip test [15]. A function $F(t)$ is unimodal with mode the region $s_m = \{(t_L, t_U) : t_L \leq t_U\}$ if it is convex in $s_L = (-\infty, t_L]$, constant in $[t_L, t_U]$, and concave in $s_U = [t_U, \infty)$. This implies the non-increasing probability density behavior when moving away from the mode. For bounded input functions $F$, $G$, let $\rho(F, G) = \max_t |F(t) - G(t)|$, and let $\mathcal{U}$ be the class of all unimodal distributions. Then the *dip statistic* of a distribution function $F$ is given by:

$$dip(F) = \min_{G \in \mathcal{U}} \rho(F, G). \tag{1}$$

In other words, the dip statistic computes the minimum among the maximum deviations observed between the cdf $F$ and the cdfs from the class of unimodal distributions. A nice property of dip is that, if $F_n$ is a sample distribution of $n$ observations from $F$, then $lim_{n \to \infty} dip(F_n) = dip(F)$. In [15] it is argued that the class of uniform distributions $U$ is the most appropriate for the null hypothesis, since its dip values are stochastically larger than other unimodal distributions, such as those having exponentially decreasing tails.

Given a vector of observations $f = \{f_i : f_i \in \mathbb{R}\}_{i=1}^{n}$, then the algorithm for performing the dip test [15] is applied on the respective empirical cdf $F_n(t) = \frac{1}{n} \sum_n I(f_i \leq t)$. It examines the $n(n\text{-}1)/2$ possible modal intervals $[t_L, t_U]$ between the sorted $n$ individual observations. For all these combinations it computes in O($n$) time the respective greatest convex minorant and the least concave majorant curves in $(min_t F_n, t_L)$ and $(t_U, max_t F_n)$, respectively. Fortunately, for a given $F_n$, the complexity of one dip computation is O($n$) [15]. The computation of the $p$-value for a unimodality test uses bootstrap samples and expresses the probability of $dip(F_n)$ being less than the dip value of a cdf $U_n^r$ of $n$ observations sampled from the U[0,1] Uniform distribution:

$$\text{P} = \#\,[dip(F_n) \leq dip(U_n^r)] \,/\, b, \; r = 1, ..., b. \tag{2}$$

The null hypothesis H$_0$ that $F_n$ is unimodal, is accepted at significance level $\alpha$ if $p$-value $> \alpha$, otherwise H$_0$ is rejected in favor of the alternative hypothesis H$_1$ which suggests multimodality.

Let a dataset $X = \{x_i : x_i \in \mathbb{R}^d\}_{i=1}^{N}$ then, in the present context, the dip test can be applied on any subset $c$, e.g. a data cluster, and more specifically on the ecdf $F_n^{(x_i)}(t) = \frac{1}{n} \sum_{x_j \in c} \{Dist(x_i, x_j) \leq t\}$ of the distances between a reference viewer $x_i$ of $c$ and the $n$ members of the set. We call the viewers that identify multimodality and vote for the set to split as *split viewers*. The dip-dist computation for a set $c$ with $n$ datapoint members is summarized as follows:

1. Compute $U_n^r$ and the respective $dip(U_n^r)$, $r=1, ..., b$, for the Uniform sample distributions.
2. Compute $F_n^{(x_i)}$ and $dip(F_n^{(x_i)})$, $i=1, ..., n$, for datapoint viewers using the sorted matrix $Dist$.
3. Estimate the $p$-values P$^{(x_i)}$, $i=1, ..., n$, based on Eq. 2 using a significance level $\alpha$ and compute the percentage of viewers identifying multimodality.

Since the ascending ordering of the rows of $Dist$, required for computing $F_n^{(x_i)}$, can be done once during offline preprocessing, and that the same $b$ samples of Uniform distribution can be used for testing all viewers, the dip-dist computation for a set with $n$ datapoints has O($bn \log n + n^2$) complexity.

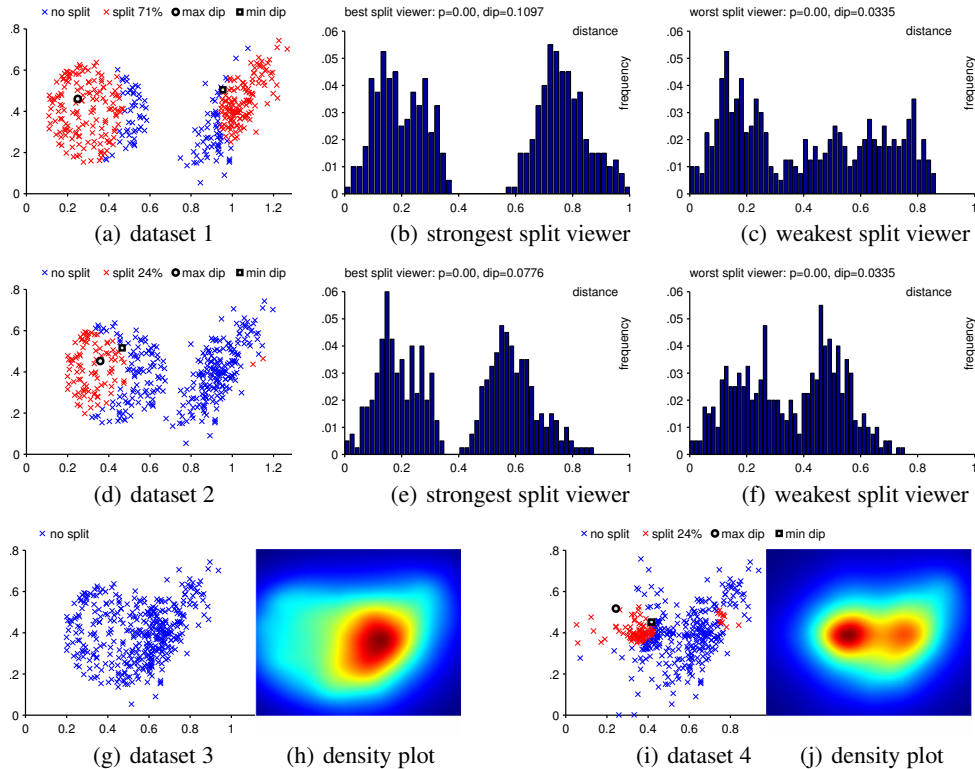

Figure 1: Application of dip-dist criterion on 2d synthetic data with two structures of 200 datapoints each. The split viewers are denoted in red color. (a) One Uniform spherical and one elliptic Gaussian structure. (b), (c) The histograms of pairwise distances of the strongest and weakest split viewer. (d) The two structures come closer; the split viewers are reduced, so does the dip value for the split viewer. (g) The two structures are no longer distinguishable as the density map in (h) shows one mode. (i) The Uniform spherical is replaced with a structure generated from a Student-t distribution.

Figure 1 illustrates an example of applying the dip-dist criterion on synthetic data. We generated a Uniform spherical and a Gaussian elliptic structure, and then constructed three different two-dimensional datasets by decreasing the distance between them. The dip test parameters are set $\alpha$=0 and $b$=1000. The histograms in each row indicate the result of the dip test. As the structures come closer, the number of viewers that observe multimodality decreases. Eventually, the structures form a unimodal distribution (Figure 1(g)), which may be visually verified from the presented density map. The fourth dataset of Figure 1(j) was created by including a structure generated by a Student-t distribution centered at the same location where the sphere is located in Figure 1(g). The respective density map shows clearly two modes, evidence that justifies why the dip-dist criterion determines multimodality with 24% of the viewers suggesting the split. More generally, if the percentage of split viewers is greater than a small threshold, e.g. 1%, we may decide that the cluster is multimodal.

## 3 The dip-means algorithm

Dip-means is an incremental clustering algorithm that combines three individual components. The first is a local search clustering technique that takes as input a model of $k$ clusters and optimizes the model parameters. For this purpose k-means is used where the cluster models are their centroids. The second, and most important, decides whether a data subset contains multiple cluster structures using the dip-dist presented in Section 2. The third component is a divisive procedure (bisecting) that, given a data subset, performs the splitting into two clusters and provides the two centers.

Dip-means methodology takes as input the dataset $X$ and two parameters for the dip-dist criterion: the significance level $\alpha$ and the percentage threshold $v_{thd}$ of cluster members that should be split viewers to decide for a division (Algorithm 1). For the sake of generality, we assume that dip-means

---
**Algorithm 1** Dip-means ($X$, $k_{init}$, $\alpha$, $v_{thd}$)
---
    **input:**   dataset $X=\{x_i\}_{i=1}^N$, the initial number of clusters $k_{init}$, a statistic significance level $\alpha$ for the unimo-
          dality test, percentage $v_{thd}$ of split viewers required for a cluster to be considered as a split candidate.
    **output:** the sets of cluster members $C=\{c_j\}_{j=1}^k$, the models $M=\{m_j\}_{j=1}^k$ with the centroid of each $c_j$ set.
    **let:**    $score$=unimodalityTest($c$, $\alpha$, $v_{thd}$) returns a score value for the cluster $c$,
          $\{C,M\}$=kmeans($X, k$) the k-means clustering, $\{C,M\}$=kmeans($X, M$) when initialized with model $M$,
          $\{m_L, m_R\}$=splitCluster($c$) that splits a cluster $c$ and returns two centers $m_L, m_R$.

1:  $k \leftarrow k_{init}$
2:  $\{C, M\} \leftarrow$ kmeans($X, k$)
3:  **do while** changes in cluster number occur
4:      **for** $j=1,\dots,k$                          *% for each cluster j*
5:          $score_j \leftarrow$ unimodalityTest($c_j$, $\alpha$, $v_{thd}$)    *% compute the score for unimodality test*
6:      **end for**
7:      **if** $\max_j(score_j) > 0$                 *% there exist split candidates*
8:          target $\leftarrow$ argmax$_j$ ($score_j$)        *% index of cluster to be splitted*
9:          $\{m_L, m_R\} \leftarrow$ splitCluster($c_{target}$)
10:         M $\leftarrow \{M-m_{target}, m_L, m_R\}$       *% replace the old centroid with the two new ones*
11:        $\{C, M\} \leftarrow$ kmeans($X, M$)          *% refine solution*
12:     **end if**
13: **end do**
14: **return** $\{C, M\}$
---

may start from any initial partition with $k_{init} \geq 1$ clusters. In each iteration, all $k$ clusters are examined for unimodality, the set of split viewers $v_j$ is found, and the respective cluster $c_j$ is characterized as *split candidate* if $|v_j|/n_j \geq v_{thd}$. In this case, a non-zero score value is assigned to each cluster being a split candidate, while zero score is assigned to clusters that do not have sufficient split viewers. Various alternatives can be employed in order to compute a score for a split candidate based on the percentage of split viewers, or even the size of clusters. In our implementation $score_j$ of a split candidate cluster $c_j$ is computed as the average value of the dip statistic of its split viewers:

$$score_j = \begin{cases} \frac{1}{|v_j|} \sum_{x_i \in v_j} dip(F^{(x_i)}), & \frac{|v_j|}{n_j} \geq v_{thd} \\ 0 & , \text{ otherwise.} \end{cases} \tag{3}$$

In order to avoid the overestimation of the real number of clusters, only the candidate with maximum score is split in each iteration. A cluster is split into two clusters using a 2-means local search approach starting from a pair of sufficiently diverse centroids $m_L$, $m_R$ inside the cluster and concerning only the datapoints of that cluster. We use a simple way to set up the initial centroids $\{m_L, m_R\} \leftarrow \{x, m-(x-m)\}$, where $x$ a cluster member selected at random and $m$ the cluster centroid. In this way $m_L$, $m_R$ lay at equal distances from $m$, though in opposite directions. The 2-means procedure can be repeated starting from different $m_L$, $m_R$ initializations in order to discover a good split. A computationally more expensive alternative could be the deterministic principal direction divisive partitioning (PDDP) [16] that splits the cluster based on the principal component. We refine the solution at the end of each iteration using k-means, which fine-tunes the model of $k+1$ clusters. The procedure terminates when no split candidates are identified among the already formed clusters.

The proposed dip-dist criterion uses only the pairwise distances, or similarities, between datapoints and not the vector representations themselves. This enables its application in kernel space $\Phi$, provided a kernel matrix $K$ with the $N \times N$ pairwise datapoint inner products, $K_{ij}=\phi(x_i)^T\phi(x_j)$. Algorithm 1 can be modified appropriately for this purpose. More specifically, *kernel dip-means* uses kernel k-means [13] as local search technique, which also implies that centroids cannot be computed in kernel space, thus each cluster is now described explicitly by the set of its members $c_j$.

In this case, since the transformed data vectors $\phi(x)$ are not available, the cluster splitting procedure could be seeded by two arbitrary cluster members. However, we propose a more efficient approach. As discussed in Section 2, the distribution of pairwise distances between a reference viewer and the members of a cluster reveals information about the multimodality of data distribution in the original space. This implies that a split of the cluster members based on their distance to a reference viewer constitutes a reasonable split in the original space, as well. To this end, we may use 2-means to split the elements of the one-dimensional similarity vector. We consider as reference split viewer the cluster member with the maximum dip value. Here, 2-means is seeded using two values located

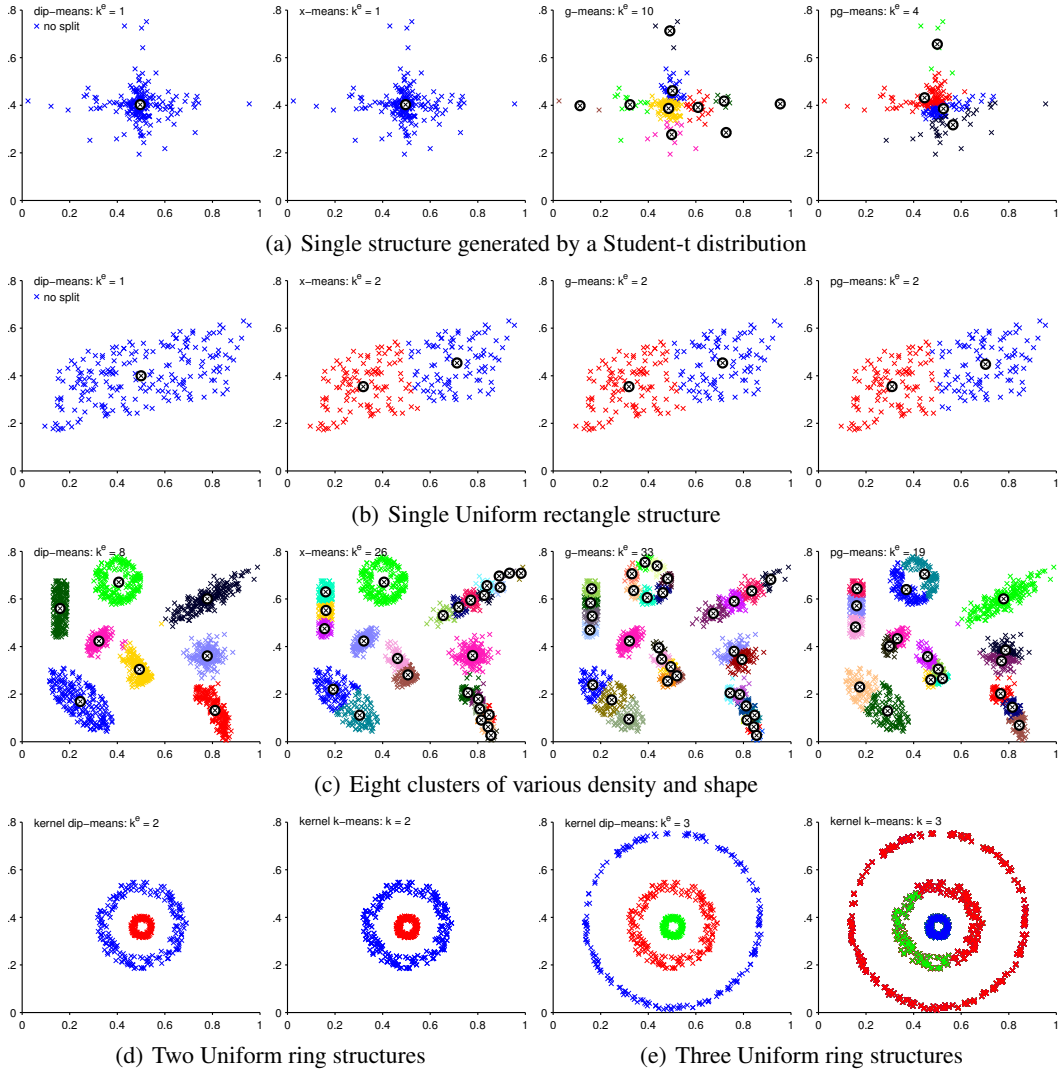

Figure 2: Clustering results on 2d synthetic unimodal cluster structures with 200 datapoints each (the centroids are marked with ⊗). (a), (b) Single cluster structures. (c) Various structure types. Based on the leftmost subfigure, it contains a Uniform rectangle (green), a sphere with increasing density at its periphery (light green), two Gaussian structures (black, pink), a Uniform ellipse (blue), a triangle denser at a corner (yellow), a Student-t (light blue), and a Uniform arbitrary shape (red). (d), (e) Non-linearly separable ring clusters (kernel-based clustering with an RBF kernel).

at opposite positions with respect to the distribution's mean. After convergence, the resulting two-way partition of the datapoints, derived by the partition of the corresponding similarity values to the selected reference split viewer, initializes a local search with kernel 2-means.

## 4 Experiments

In our evaluation we compare the proposed dip-means method with x-means [1], g-means [4] and pg-means [5] that are closely related to present work. In all compared methods we start with a single cluster ($k_{init}$=1) and i) at each iteration one cluster is selected for a bisecting split, ii) 10 split trials are performed with 2-means initialized with the simple technique described in Section 3, and the split with lower clustering error (the sum of squared differences between cluster centers and their assigned datapoints) is kept, iii) the refinement is applied after each iteration on all $k$+1 clusters. Hence, only the statistical test that decides whether to stop splitting differs in each case. Exception

Table 1: Results for synthetic datasets with fixed $k^*$=20 clusters with 200 datapoints in each cluster.

| Methods | Case 1, $d$=4 | | | Case 1, $d$=16 | | | Case 1, $d$=32 | | |
|---|---|---|---|---|---|---|---|---|---|
| | $k^e$ | ARI | VI | $k^e$ | ARI | VI | $k^e$ | ARI | VI |
| dip-means | 20.0±0.0 | 1.00±0.0 | 0.00±0.0 | 20.0±0.0 | 1.00±0.0 | 0.00±0.0 | 20.0±0.0 | 1.00±0.0 | 0.00±0.0 |
| x-means | 7.3±9.3 | 0.30±0.5 | 2.07±1.3 | 28.6±7.8 | 0.88±0.1 | 0.27±0.2 | 31.3±5.6 | 0.84±0.1 | 0.36±0.2 |
| g-means | 20.3±0.5 | 0.99±0.0 | 0.01±0.0 | 20.3±0.5 | 0.99±0.0 | 0.01±0.0 | 20.5±0.6 | 0.99±0.0 | 0.02±0.0 |
| pg-means | 19.2±2.5 | 0.90±0.1 | 0.16±0.2 | 19.0±0.9 | 0.95±0.1 | 0.07±0.1 | 3.2±5.1 | 0.09±0.2 | 2.62±0.9 |

| Methods | Case 2, $d$=4 | | | Case 2, $d$=16 | | | Case 2, $d$=32 | | |
|---|---|---|---|---|---|---|---|---|---|
| | $k^e$ | ARI | VI | $k^e$ | ARI | VI | $k^e$ | ARI | VI |
| dip-means | 20.0±0.0 | 0.99±0.0 | 0.05±0.0 | 20.0±0.0 | 0.99±0.0 | 0.02±0.0 | 20.0±0.0 | 0.99±0.0 | 0.01±0.0 |
| x-means | 24.8±39. | 0.26±0.4 | 2.26±1.1 | 80.1±15. | 0.75±0.1 | 0.75±0.2 | 71.6±14. | 0.75±0.1 | 0.66±0.2 |
| g-means | 79.2±22. | 0.77±0.1 | 0.70±0.2 | 105.9±30. | 0.83±0.1 | 0.66±0.2 | 133.6±42. | 0.83±0.1 | 0.72±0.2 |
| pg-means | 14.2±4.7 | 0.67±0.2 | 0.65±0.5 | 10.4±3.4 | 0.30±0.2 | 1.26±0.5 | 4.0±1.5 | 0.06±0.1 | 2.40±0.2 |

is the pg-means method that uses EM for local search and does not rely on cluster splitting to add a new cluster. We use the method exactly as presented in [5]. For the kernel-based experiments we use the necessary modifications described at the end of Section 3 and compare with kernel k-means [13]. The parameters of the dip-dist criterion are set as $\alpha$=0 for significance level of dip test and $b$=1000 for the number of bootstraps. We consider as split candidates the clusters having at least $v_{thd}$=1% split viewers. These values were fixed in all experiments. For both g-means and pg-means we set the significance level $\alpha$=0.001, while we use 12 random projections for the latter. In order to compare the ground truth labeling and the grouping produced by clustering, we utilize the Variation of Information (VI) [17] metric and the Adjusted Rand Index (ARI) [18]. Better clustering is indicated by lower values of VI and higher for ARI.

We first provide clustering results for synthetic 2d datasets in Figure 2 ($k^e$ denotes the estimated number of clusters). In Figures 2(a), (b), we provide two indicative examples of single cluster structures. X-means decides correctly for the structure generated from Student-t distribution, but overfits in the Uniform rectangle case, while the other two methods overfit in both cases. In the multicluster dataset of Figure 2(c), dip-means successfully discovers all clusters, in contrast to the other methods that significantly overestimate. To test the kernel dip-means extension, we created two 2d synthetic dataset containing two and three Uniform ring structures and we used an RBF kernel to construct the kernel matrix $K$. It is clear that x-means, g-means, and pg-means are not applicable in this case. Thus we present in Figures 2(d), 2(e) the results using kernel dip-means and also the best solution from 50 randomly initialized runs of kernel k-means with the true number of clusters. As we may observe, dip-means estimates the true number of clusters and finds the optimal grouping of datapoints in both cases, whereas kernel k-means fails in the three ring case. Furthermore, we created synthetic datasets with true number $k^*$=20 clusters, with 200 datapoints each, in $d$= 4, 16, 32 dimensions with low separation [19]. Two cases were considered: 1) Gaussian mixtures of varying eccentricity, and 2) datasets with various cluster structures, i.e. Gaussian (40%), Student-t (20%), Uniform ellipses (20%) or Uniform rectangles (20%). For each case and dimensions, we generated 30 datasets to test the methods. As the results in Table 1 indicate, dip-means provides excellent clustering performance in all cases and estimates accurately the true number of clusters. Moreover, it performs remarkably better than the other methods, especially for the datasets of Case 2.

Two real-world datasets were also used, where the provided class labels were considered as ground truth. Handwritten Pendigits (UCI) [20] contains 16 dimensional vectors, each one representing a digit from 0-9 written by a human subject. The data provide a training $PD_{tr}$ and a testing set $PD_{te}$ with 7494 and 3498 instances, respectively. We also consider two subsets that contain the digits $\{0, 2, 4\}$ ($PD3_{tr}$ and $PD3_{te}$) and $\{3, 6, 8, 9\}$ ($PD4_{tr}$ and $PD4_{te}$). We do not apply any preprocessing. Coil-100 is the second dataset [21], which contains 72 images taken from different angles for each one of the 100 included objects. We used tree subsets Coil3, Coil4, Coil5, with images from 3, 4 and 5 objects, respectively. SIFT descriptors [22] are first extracted from the greyscale images that are finally represented by the *Bag of Visual Words* model using 1000 visual words. As reported in Table 2, dip-means correctly discovers the number of clusters for the subsets of Pendigits, while providing a reasonable underestimation $k^e$ near the optimal for the full datasets $PD10_{tr}$ and $PD10_{te}$. Apart from the excessive overfitting of x-means and g-means, pg-means seems to concludes in overestimated $k^e$. In the high dimensional and sparse space of the considered Coil subsets, x-means

Table 2: Clustering results for real-world data. Bold indicates best values.

| | PD3$_{te}$ ($k^*$=3) | | | PD4$_{te}$ ($k^*$=4) | | | PD10$_{te}$ ($k^*$=10) | | |
|---|---|---|---|---|---|---|---|---|---|
| Methods | $k^e$ | ARI | VI | $k^e$ | ARI | VI | $k^e$ | ARI | VI |
| dip-means | 3 | **0.879** | **0.332** | 4 | **0.626** | **0.545** | 7 | 0.343 | **1.587** |
| x-means | 155 | 0.031 | 3.792 | 194 | 0.039 | 3.723 | 515 | 0.041 | 3.825 |
| g-means | 21 | 0.226 | 1.800 | 36 | 0.209 | 2.049 | 73 | 0.295 | 1.961 |
| pg-means | 4 | 0.835 | 0.359 | 10 | 0.576 | 0.954 | 13 | **0.447** | 1.660 |
| | PD3$_{tr}$ ($k^*$=3) | | | PD4$_{tr}$ ($k^*$=4) | | | PD10$_{tr}$ ($k^*$=10) | | |
| Methods | $k^e$ | ARI | VI | $k^e$ | ARI | VI | $k^e$ | ARI | VI |
| dip-means | 3 | **0.963** | **0.116** | 4 | **0.522** | 0.841 | 9 | 0.435 | **1.452** |
| x-means | 288 | 0.018 | 4.378 | 381 | 0.020 | 4.372 | 942 | 0.024 | 4.387 |
| g-means | 52 | 0.106 | 2.641 | 58 | 0.143 | 2.464 | 149 | 0.160 | 2.605 |
| pg-means | 5 | 0.655 | 0.740 | 8 | 0.439 | 1.320 | 14 | **0.494** | 1.504 |
| | Coil3 ($k^*$=3) | | | Coil4 ($k^*$=4) | | | Coil5 ($k^*$=5) | | |
| Methods | $k^e$ | ARI | VI | $k^e$ | ARI | VI | $k^e$ | ARI | VI |
| dip-means | 3 | **1.000** | **0.000** | 5 | **0.912** | **0.173** | 4 | **0.772** | **0.308** |
| x-means | 8 | 0.499 | 0.899 | 11 | 0.499 | 0.951 | 15 | 0.601 | 0.907 |
| g-means | 7 | 0.669 | 0.650 | 12 | 0.502 | 0.977 | 18 | 0.434 | 1.204 |

and g-means provide more reasonable $k^e$ estimations, but still overestimations. An explanation for this behavior is that they discover smaller groups of similar images, i.e. images taken from close angles to the same object, but fail to unify the subclusters at higher level. Note also that we did not manage to test pg-means in Coil-100 subsets, since covariance matrices were not positive definite. The superiority of dip-means is also indicated by the reported values for ARI and VI measures.

## 5  Conclusions

We have presented a novel approach for testing whether multiple cluster structures are present in a set of data objects (e.g. a data cluster). The proposed *dip-dist criterion* checks for unimodality of the empirical data density distribution, thus it is much more general compared to alternatives that test for Gaussianity. Dip-dist uses a statistical hypothesis test, namely Hartigans' dip test, in order to verify unimodality. If a data object of the set is considered as a *viewer*, then the dip test can be applied on the one-dimensional distance (or similarity) vector with components the distances between the viewer and the members of the same set. We exploit the idea that the observation of multimodality in the distribution of distances indicates multimodality of the original data distribution. By considering all the data objects of the set as individual viewers and by combining the respective results of the test, the presence of multiple cluster structures in the set can be determined.

We have also proposed a new incremental clustering algorithm called *dip-means*, that incorporates dip-dist criterion in order to decide for cluster splitting. The procedure starts with one cluster, it iteratively splits the cluster indicated by dip-dist as more probable to contain multiple cluster structures, and terminates when no new cluster split is suggested. By taking advantage of the fact that dip-dist utilizes only information about the distances between data objects, we have modified appropriately the main algorithm to propose *kernel dip-means* which can be applied in kernel space.

The proposed method is fast, easy to implement, and works very well under a fixed parameter setting. The reported clustering results indicate that dip-means can provide reasonable estimates of the number of clusters, and produce meaningful clusterings in both dataset types in a variety of artificial and real datasets. Apart from testing the method in real-world applications, there are several ways to improve the implementation details of the method, especially the kernel-based version. We also plan to test its effectiveness in other settings, such as online clustering of stream data.

### Acknowledgments

We thank Prof. Greg Hamerly for providing his code for pg-means. The described work is supported partially and co-financed by the European Regional Development Fund (ERDF) (2007-2013) of the European Union and National Funds (Operational Programme "Competitiveness and Entrepreneurship" (OPCE II), ROP ATTICA), under the Action "SYNERGASIA (COOPERATION) 2009".

## References

[1] D. Pelleg and Andrew Moore. X-means: extending k-means with efficient estimation of the number of clusters. *International Conference on Machine Learning (ICML)*, pp. 727-734, 2000.

[2] R.E. Kass and L. Wasserman. A reference Bayesian test for nested hypotheses and its relationship to the Schwarz criterion. *Journal of the American Statistical Association*, 90(431), pp. 928-934, 1995.

[3] X. Hu and L. Xu. A comparative study of several cluster number selection criteria. In *J. Liu et al.(eds.) Intelligent Data Engineering and Automated Learning*, pp. 195–202, Springer, 2003.

[4] G. Hamerly and C. Elkan. Learning the k in k-means. *Advances in Neural Information Processing Systems (NIPS)*, pp. 281-288, 2003.

[5] Y. Feng and G. Hamerly. PG-means: learning the number of clusters in data. *Advances in Neural Information Processing Systems (NIPS)*, pp. 393–400, 2006.

[6] K. Kurihara and M. Welling. Bayesian k-means as a maximization-expectation algorithm. *Neural Computation*, 21(4), pp. 1145–1172, 2009.

[7] R. Tibshirani, G. Walther and T. Hastie. Estimating the number of clusters in a dataset via the Gap statistic. *Journal of the Royal Statistical Society B*, 63, pp. 411-423, 2001.

[8] L. Zelnik-Manor and P. Perona. Self-tuning spectral clustering. *Advances in Neural Information Processing Systems (NIPS)*, pp. 1601–1608, 2004.

[9] T. Shi, M. Belkin and B. Yu. Data Spectroscopy: eigenspaces of convolution operators and clustering. *The Annals of Statistics*, 37(6B), pp. 3960–3984, 2009.

[10] E. Levine and E. Domany. Resampling method for unsupervised estimation of cluster validity. *Neural Computation*, 13(11), pp. 2573-2593, 2001.

[11] Robert Tibshirani and G. Walther. Cluster validation by prediction strength. *Computational & Graphical Statistics*, 14(3), pp. 511-528, 2005.

[12] T. Lange, V. Roth, Mikio L. Braun, and J.M. Buhmann. Stability-based validation of clustering solutions. *Neural Computation*, 16(6), pp. 1299-1323, 2004.

[13] I.S. Dhillon, Y. Guan and B. Kulis. Kernel k-means: spectral clustering and normalized cuts. *International Conference on Knowledge Discovery and Data Mining (SIGKDD)*, pp. 551–556, 2004.

[14] B.W. Silverman. Using Kernel density estimates to investigate multimodality. *Journal of Royal Statistic Society B*, 43(1), pp. 97-99, 1981.

[15] J.A. Hartigan and P. M. Hartigan. The dip test of unimodality. *The Annals of Statistics*, 13(1), pp. 70-84, 1985.

[16] D.L. Boley. Principal direction divisive partitioning. *Data Mining and Knowledge Discovery*, 2(4), pp. 344, 1998.

[17] M. Meila. Comparing clusterings – an information based distance. *Multivariate Analysis*, 98(5), pp. 873-895, 2007.

[18] L. Hubert and P. Arabie. Comparing partitions. *Journal of Classification*, 2(1), pp. 193-218, 1985.

[19] J.J. Verbeek, N. Vlassis, and B. Kro"se. Efficient Greedy Learning of Gaussian Mixture Models. *Neural Computation*, 15(2), pp. 469-485, 2003.

[20] A. Asuncion and D. Newman. UCI Machine Learning Repository. *University of California at Irvine*, Irvine, CA, 2007. Available online: http://www.ics.uci.edu/ mlearn/MLRepository.html

[21] S.A. Nene, S.K. Nayar and H. Murase. Columbia Object Image Library (COIL-100). *Technical Report CUCS-006-96*, February 1996.

[22] D. Lowe. Distinctive image features from scale-invariant keypoints. *Journal of Computer Vision*, 60, pp. 91-110, 2004.

